# Extracting Dynamical Structure Embedded in Neural Activity

**Byron M. Yu**[1], **Afsheen Afshar**[1,2], **Gopal Santhanam**[1],
**Stephen I. Ryu**[1,3], **Krishna V. Shenoy**[1,4]
[1]Department of Electrical Engineering, [2]School of Medicine, [3]Department of
Neurosurgery, [4]Neurosciences Program, Stanford University, Stanford, CA 94305
{byronyu,afsheen,gopals,seoulman,shenoy}@stanford.edu

**Maneesh Sahani**
Gatsby Computational Neuroscience Unit, UCL
London, WC1N 3AR, UK
maneesh@gatsby.ucl.ac.uk

## Abstract

Spiking activity from neurophysiological experiments often exhibits dynamics beyond that driven by external stimulation, presumably reflecting the extensive recurrence of neural circuitry. Characterizing these dynamics may reveal important features of neural computation, particularly during internally-driven cognitive operations. For example, the activity of premotor cortex (PMd) neurons during an instructed delay period separating movement-target specification and a movement-initiation cue is believed to be involved in motor planning. We show that the dynamics underlying this activity can be captured by a low-dimensional non-linear dynamical systems model, with underlying recurrent structure and stochastic point-process output. We present and validate latent variable methods that simultaneously estimate the system parameters and the trial-by-trial dynamical trajectories. These methods are applied to characterize the dynamics in PMd data recorded from a chronically-implanted 96-electrode array while monkeys perform delayed-reach tasks.

## 1 Introduction

At present, the best view of the activity of a neural circuit is provided by multiple-electrode extracellular recording technologies, which allow us to simultaneously measure spike trains from up to a few hundred neurons in one or more brain areas during each trial. While the resulting data provide an extensive picture of neural spiking, their use in characterizing the fine timescale dynamics of a neural circuit is complicated by at least two factors. First, extracellularly captured action potentials provide only an occasional view of the process from which they are generated, forcing us to interpolate the evolution of the circuit between the spikes. Second, the circuit activity may evolve quite differently on different trials that are otherwise experimentally identical.

The usual approach to handling both problems is to average responses from different trials, and study the evolution of the peri-stimulus time histogram (PSTH). There is little alternative to this approach when recordings are made one neuron at a time, even when the dynamics of the system are the subject of study. Unfortunately, such averaging can obscure important internal features of the response. In many experiments, stimulus events provide the trigger for activity, but the resulting time-course of the response is internally regulated and may not be identical on each trial. This is especially important during cognitive processing such as decision making or motor planning. In this case, the PSTH may not reflect the true trial-by-trial dynamics. For example, a sharp change in firing rate that occurs with varying latency might appear as a slow smooth transition in the average response.

An alternative approach is to adopt latent variable methods and to identify a hidden dynamical system that can summarize and explain the simultaneously-recorded spike trains. The central idea is that the responses of different neurons reflect different views of a common dynamical process in the network, whose effective dimensionality is much smaller than the total number of neurons in the network. While the underlying state trajectory may be slightly different on each trial, the commonalities among these trajectories can be captured by the network's parameters, which are shared across trials. These parameters define how the network evolves over time, as well as how the observed spike trains relate to the network's state at each time point.

Dimensionality reduction in a latent dynamical model is crucial and yields benefits beyond simple noise elimination. Some of these benefits can be illustrated by a simple physical example. Consider a set of noisy video sequences of a bouncing ball. The trajectory of the ball may not be identical in each sequence, and so simply averaging the sequences together would provide little information about the dynamics. Independently smoothing the dynamics of each pixel might identify a dynamical process; however, correctly rejecting noise might be difficult, and in any case this would yield an inefficient and opaque representation of the underlying physical process. By contrast, a hidden dynamical system account could capture the video sequence data using a low-dimensional latent variable that represented only the ball's position and momentum over time, with dynamical rules that captured the physics of ballistics and elastic collision. This representation would exploit shared information from all pixels, vastly simplifying the problem of noise rejection, and would provide a scientifically useful depiction of the process.

The example also serves to illustrate the two broad benefits of this type of model. The first is to obtain a low dimensional summary of the dynamical trajectory in any one trial. Besides the obvious benefits of denoising, such a trajectory can provide an invaluable representation for prediction of associated phenomena. In the video sequence example, predicting the loudness of the sound on impact might be easy given the estimate of the ball's trajectory (and thus its speed), but would be difficult from the raw pixel trajectories, even if denoised. In the neural case, behavioral variables such as reaction time might similarly be most easily predicted from the reconstructed trajectory. The second broad goal is systems identification: learning the rules that govern the dynamics. In the video example this would involve discovery of various laws of physics, as well as parameters describing the ball such as its coefficient of elasticity. In the neural case this would involve identifying the structure of dynamics available to the circuit: the number and relationship of attractors, appearance of oscillatory limit cycles and so on.

The use of latent variable models with hidden dynamics for neural data has, thus far, been limited. In [1], [2], small groups of neurons in the frontal cortex were modeled using hidden Markov models, in which the latent dynamical system is assumed to transition between a set of discrete states. In [3], a state space model with linear hidden dynamics and point-process outputs was applied to simulated data. However, these restricted latent models cannot capture the richness of dynamics that recurrent networks exhibit. In particular, systems that converge toward point or line attractors, exhibit limit cycle oscillations, or

even transition into chaotic regimes have long been of interest in neural modeling. If such systems are relevant to real neural data, we must seek to identify hidden models capable of reflecting this range of behaviors.

In this work, we consider a latent variable model having (1) hidden underlying recurrent structure with continuous-valued states, and (2) Poisson-distributed output spike counts (conditioned on the state), as described in Section 2. Inference and learning for this nonlinear model are detailed in Section 3. The methods developed are applied to a delayed-reach task described in Section 4. Evidence of motor preparation in PMd is given in Section 5. In Section 6, we characterize the neural dynamics of motor preparation on a trial-by-trial basis.

## 2 Hidden non-linear dynamical system

A useful dynamical system model capable of expressing the rich behavior expected of neural systems is the recurrent neural network (RNN) with Gaussian perturbations

$$\mathbf{x}_t \mid \mathbf{x}_{t-1} \sim \mathcal{N}\left(\psi(\mathbf{x}_{t-1}), \mathsf{Q}\right) \tag{1}$$

$$\psi(\mathbf{x}) = (1-k)\mathbf{x} + kWg(\mathbf{x}), \tag{2}$$

where $\mathbf{x}_t \in \mathbb{R}^{p \times 1}$ is the vector of the node values in the recurrent network at time $t \in \{1, \ldots, T\}$, $W \in \mathbb{R}^{p \times p}$ is the connection weight matrix, $g$ is a non-linear activation function which acts element-by-element on its vector argument, $k \in \mathbb{R}$ is a parameter related to the time constant of the network, and $\mathsf{Q} \in \mathbb{R}^{p \times p}$ is a covariance matrix. The initial state is Gaussian-distributed

$$\mathbf{x}_0 \sim \mathcal{N}\left(\mathbf{p}_0, \mathsf{V}_0\right), \tag{3}$$

where $\mathbf{p}_0 \in \mathbb{R}^{p \times 1}$ and $\mathsf{V}_0 \in \mathbb{R}^{p \times p}$ are the mean vector and covariance matrix, respectively.

Models of this class have long been used, albeit generally without stochastic perturbation, to describe the dynamics of neuronal responses (e.g., [4]). In this classical view, each node of the network represents a neuron or a column of neurons. Our use is more abstract. The RNN is chosen for the range of dynamics it can exhibit, including convergence to point or surface attractors, oscillatory limit cycles, or chaotic evolution; but each node is simply an abstract dimension of latent space which may couple to many or all of the observed neurons.

The output distribution is given by a generalized linear model that describes the relationship between all nodes in the state $\mathbf{x}_t$ and the spike count $y_t^i \in \mathbb{R}$ of neuron $i \in \{1, \ldots, q\}$ in the $t$th time bin

$$y_t^i \mid \mathbf{x}_t \sim \text{Poisson}\left(h\left(\mathbf{c}^i \cdot \mathbf{x}_t + d^i\right) \Delta\right), \tag{4}$$

where $\mathbf{c}^i \in \mathbb{R}^{p \times 1}$ and $d^i \in \mathbb{R}$ are constants, $h$ is a link function mapping $\mathbb{R} \to \mathbb{R}_+$, and $\Delta \in \mathbb{R}$ is the time bin width. We collect the spike counts from all $q$ simultaneously-recorded physical neurons into a vector $\mathbf{y}_t \in \mathbb{R}^{q \times 1}$, whose $i$th element is $y_t^i$. The choice of the link functions $g$ and $h$ is discussed in Section 3.

## 3 Inference and Learning

The Expectation-Maximization (EM) algorithm [5] was used to iteratively (1) *infer* the underlying hidden state trajectories (i.e., recover a distribution over the hidden sequence $\{\mathbf{x}\}_1^T$ corresponding to the observations $\{\mathbf{y}\}_1^T$), and (2) *learn* the model parameters (i.e., estimate $\theta = \{W, \mathsf{Q}, k, \mathbf{p}_0, \mathsf{V}_0, \{\mathbf{c}^i\}, \{d^i\}\}$), given only a set of observation sequences.

Inference (the E-step) involves computing or approximating $P\left(\{\mathbf{x}\}_1^T \mid \{\mathbf{y}\}_1^T, \theta_k\right)$ for each sequence, where $\theta_k$ are the parameter estimates at the $k$th EM iteration. A variant of the Extended Kalman Smoother (EKS) was used to approximate these joint smoothed state posteriors. As in the EKS, the non-linear time-invariant state system (1)-(2) was transformed into a linear time-variant sytem using local linearization. The difference from EKS arises in the measurement update step of the forward pass

$$P\left(\mathbf{x}_t \mid \{\mathbf{y}\}_1^t\right) \propto P\left(\mathbf{y}_t \mid \mathbf{x}_t\right) P\left(\mathbf{x}_t \mid \{\mathbf{y}\}_1^{t-1}\right). \tag{5}$$

Because $P\left(\mathbf{y}_t \mid \mathbf{x}_t\right)$ is a product of Poissons rather than a Gaussian, the filtered state posterior $P\left(\mathbf{x}_t \mid \{\mathbf{y}\}_1^t\right)$ cannot be easily computed. Instead, as in [3], we approximated this posterior with a Gaussian centered at the mode of $\log P\left(\mathbf{x}_t \mid \{\mathbf{y}\}_1^t\right)$ and whose covariance is given by the negative inverse Hessian of the log posterior at that mode. Certain choices of $h$, including $e^z$ and $\log\left(1 + e^z\right)$, lead to a log posterior that is strictly concave in $\mathbf{x}_t$. In these cases, the unique mode can easily be found by Newton's method.

Learning (the M-step) requires finding the $\theta$ that maximizes $E\left[\log P\left(\{\mathbf{x}\}_1^T, \{\mathbf{y}\}_1^T \mid \theta\right)\right]$, where the expectation is taken over the posterior state distributions found in the E-step. Note that, for multiple sequences that are independent conditioned on $\theta$, we use the *sum* of expectations over all sequences. Because the posterior state distributions are approximated as Gaussians in the E-step, the above expectation is a Gaussian integral that involves non-linear functions $g$ and $h$ and cannot be computed analytically in general. Fortunately, this high-dimensional integral can be reduced to many one-dimensional Gaussian integrals, which can be accurately and reasonably efficiently approximated using Gaussian quadrature [6], [7].

We found that setting $g$ to be the error function

$$g(z) = \frac{2}{\sqrt{\pi}} \int_0^z e^{-t^2} dt \tag{6}$$

made many of the one-dimensional Gaussian integrals involving $g$ analytically tractable. Those that were not analytically tractable were approximated using Gaussian quadrature. The error function is one of a family of sigmoid activation functions that yield similar behavior in a RNN.

If $h$ were chosen to be a simple exponential, all the Gaussian integrals involving $h$ could be computed exactly. Unfortunately, this exponential mapping would distort the relationship between perturbations in the latent state (whose size is set by the covariance matrix $\mathbf{Q}$) and the resulting fluctuations in firing rates. In particular, the size of firing-rate fluctuations would grow *exponentially* with the mean, an effect that would then add to the usual linear increase in spike-count variance that comes from the Poisson output distribution. Since neural firing does not show such a severe scaling in variability, such a model would fit poorly. Therefore, to maintain more even firing-rate fluctuations, we instead take

$$h(z) = \log\left(1 + e^z\right). \tag{7}$$

The corresponding Gaussian integrals must then be approximated by quadrature methods. Regardless of the forms of $g$ and $h$ chosen, numerical Newton methods are needed for maximization with respect to $\{\mathbf{c}^i\}$ and $\{d^i\}$.

The main drawback of these various approximations is that the overall observation likelihood is no longer guaranteed to increase after each EM iteration. However, in our simulations, we found that sensible results were often produced. As long as the variances of the posterior state distribution did not diverge, the output distributions described by the learned model closely approximated those of the actual model that generated the simulated data.

## 4   Task and recordings

We trained a rhesus macaque monkey to perform delayed center-out reaches to visual targets presented on a fronto-parallel screen. On a given trial, the peripheral target was presented at one of eight radial locations (30, 70, 110, 150, 190, 230, 310, 350°) 10 cm away, as shown in Figure 1. After a pseudo-randomly chosen delay period of 200, 750, or 1000 ms, the target increased in size as the go cue and the monkey reached to the target. A 96-channel silicon electrode array (Cyberkinetics, Inc.) was implanted straddling PMd and motor cortex (M1). Spike sorting was performed offline to isolate 22 single-neuron and 109 multi-neuron units.

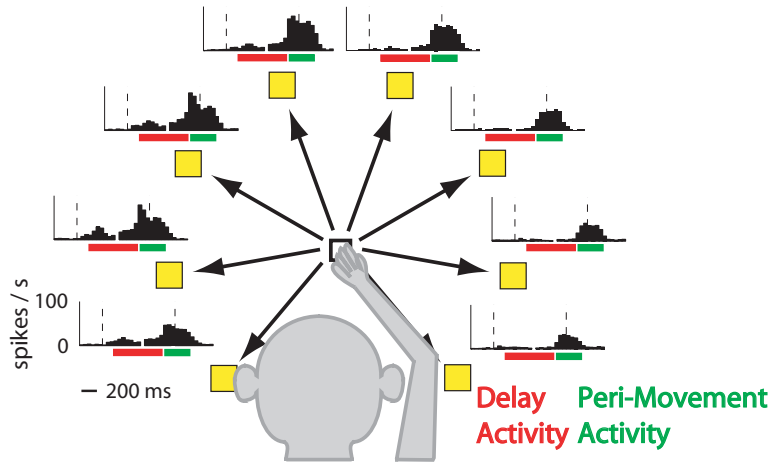

Figure 1: Delayed reach task and average action potential (spike) emission rate from one representative unit. Activity is arranged by target location. Vertical dashed lines indicate peripheral reach target onset (left) and movement onset (right).

## 5   Motor preparation in PMd

Motor preparation is often studied using the "instructed delay" behavioral paradigm, as described in Section 4, where a variable-length "planning" period temporally separates an instruction stimulus from a go cue [8]–[13]. Longer delay periods typically lead to shorter reaction times (RT, defined as time between go cue and movement onset), and this has been interpreted as evidence for a motor preparation process that takes time [11], [12], [14], [15]. In this view, the delay period allows for motor preparation to complete prior to the go cue, thus shortening the RT.

Evidence for motor preparation at the neural level is taken from PMd (and, to a lesser degree, M1), where neurons show sustained activity during the delay period (Figure 1, delay activity) [8]–[10]. A number of findings support the hypothesis that such activity is related to motor preparation. First, delay period activity typically shows tuning for the instruction (i.e., location of reach target; note that the PMd neuron in Figure 1 has greater delay activity before leftward than before rightward reaches), consistent with the idea that something specific is being prepared [8], [9], [11], [13]. Second, in the absence of a delay period, a brief burst of similarly-tuned activity is observed during the RT interval, consistent with the idea that motor preparation is taking place at that time [12].

Third, we have recently reported that firing rates *across trials* to the same reach target become more consistent as the delay period progresses [16]. The variance of firing rate,

measured across trials, divided by mean firing rate (similar to the Fano factor) was computed for each unit and each time point. Averaged across 14 single- and 33 multi-neuron units, we found that this Normalized Variance (NV) declined 24% (t-test, p $<10^{-10}$) from 200 ms before target onset to the median time of the go cue. This decline spanned ~119 ms just after target onset and appears to, at least roughly, track the time-course of motor preparation.

The NV may be interpreted as a signature of the approximate degree of motor preparation yet to be accomplished. Shortly after target onset, firing rates are frequently far from their mean. If the go cue arrives then, it will take time to correct these "errors" and RTs will therefore be longer. By the time the NV has completed its decline, firing rates are consistently near their mean (which we presume is near an "optimal" configuration for the impending reach), and RTs will be shorter if the go cue arrives then. This interpretation assumes that there is a limit on how quickly firing rates can converge to their ideal values (a limit on how quickly the NV can drop) such that a decline during the delay period saves time later. The NV was found to be lower at the time of the go cue for trials with shorter RTs than those with longer RTs [16].

The above data strongly suggest that the network underlying motor preparation exhibits rich dynamics. Activity is initially variable across trials, but appears to settle during the delay period. Because the RNN (1)-(2) is capable of exhibiting such dynamics and may underly motor preparation, we sought to identify such a dynamical system in delay activity.

## 6 Results and discussion

The NV reveals an average process of settling by measuring the convergence of firing across different trials. However, it provides little insight into the course of motor planning on a single trial. A gradual fall in trial-to-trial variance might reflect a gradual convergence on each trial, or might reflect rapid transitions that occur at different times on different trials. Similarly, all the NV tells us about the dynamic properties of the underlying network is the basic fact of convergence from uncontrolled initial conditions to a consistent pre-movement preparatory state. The structure of any underlying attractors and corresponding basins of attraction is unobserved. Furthermore, the NV is first computed per-unit and averaged across units, thus ignoring any structure that may be present in the correlated firing of units on a given trial. The methods presented here are well-suited to extending the characterization of this settling process.

We fit the dynamical system model (1)–(4) with three latent dimensions ($p = 3$) to training data, consisting of delay activity preceding 70 reaches to the same target (30°). Spike counts were taken in non-overlapping $\Delta = 20$ ms bins at 20 ms time steps from 50 ms after target onset to 50 ms after the go cue. Then, the fitted model parameters were used to infer the latent space trajectories for 146 test trials, which are plotted in Figure 2. Despite the trial-to-trial variability in the delay period neural responses, the state evolves along a characteristic path on each trial. It could have been that the neural variability across trials would cause the state trajectory to evolve in markedly different ways on different trials. Even with the characteristic structure, the state trajectories are not all identical, however. This presumably reflects the fact that the motor planning process is internally-regulated, and its timecourse may differ from trial to trial, even when the presented stimulus (in this case, the reach target) is identical. How these timecourses differ from trial to trial would have been obscured had we combined the neural data across trials, as with the NV in Section 5.

Is this low-dimensional description of the system dynamics adequate to describe the firing of all 131 recorded units? We transformed the inferred latent trajectories into trial-by-trial

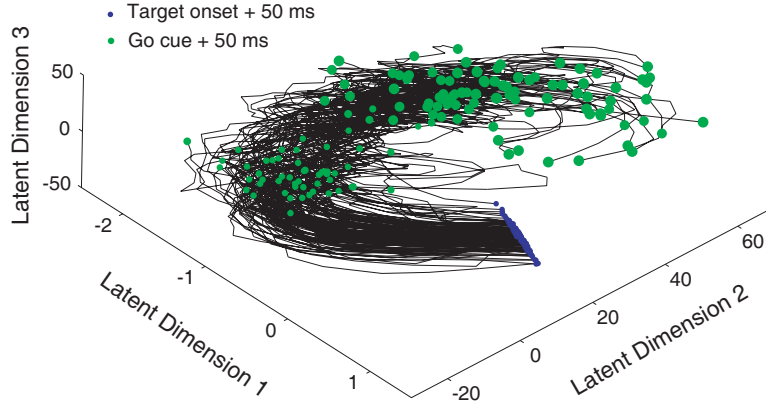

Figure 2: Inferred modal state trajectories in latent ($\mathbf{x}$) space for 146 test trials. Dots indicate 50 ms after target onset (blue) and 50 ms after the go cue (green). The radius of the green dots is logarithmically-related to delay period length (200, 750, or 1000ms).

inhomogeneous firing rates using the output relationship from (4)

$$\lambda_t^i = h\left(\mathbf{c}^i \cdot \mathbf{x}_t + d^i\right), \tag{8}$$

where $\lambda_t^i$ is the imputed firing rate of the $i$th unit at the $t$th time bin. Figure 3 shows the imputed firing rates for 15 representative units overlaid with empirical firing rates obtained by directly averaging raw spike counts across the same test trials. If the imputed firing rates truly reflect the rate functions underlying the observed spikes, then the mean behavior of the imputed firing rates should track the empirical firing rates. On the other hand, if the latent system were inadequate to describe the activity, we should expect to see dynamical features in the empirical firing that could not be captured by the imputed firing rates. The strong agreement observed in Figure 3 and across all 131 units suggests that this simple dynamical system is indeed capable of capturing significant components of the dynamics of this neural circuit. We can view the dyamical system approach adopted in this work as a form of *non-linear dynamical embedding* of point-process data. This is in contrast to most current embedding algorithms that rely on continuous data. Figure 2 effectively represents a three-dimensional manifold in the space of firing rates along which the dynamics unfold.

Beyond the agreement of imputed means demonstrated by Figure 3, we would like to directly test the fit of the model to the neural spike data. Unfortunately, current goodness-of-fit methods for spike trains, such as those based on time-rescaling [17], cannot be applied directly to latent variable models. The difficulty arises because the average trajectory obtained from marginalizing over the latent variables in the system (by which we might hope to rescale the inter-spike intervals) is not designed to provide an accurate estimate of the trial-by-trial firing rate functions. Instead, each trial must be described by a distinct trajectory in latent space, which can only be inferred after observing the spike trains themselves. This could lead to overfitting. We are currently exploring extensions to the standard methods which infer latent trajectories using a subset of recorded neurons, and then test the quality of firing-rate predictions for the remaining neurons. In addition, we plan to compare models of different latent dimensionalities; here, the latent space was arbitrarily chosen to be three-dimensional. To validate the learned latent space and inferred trajectories, we would also like to relate these results to trial-by-trial behavior. In particular, given the evidence from Section 5, how "settled" the activity is at the time of the go cue should be predictive of RT.

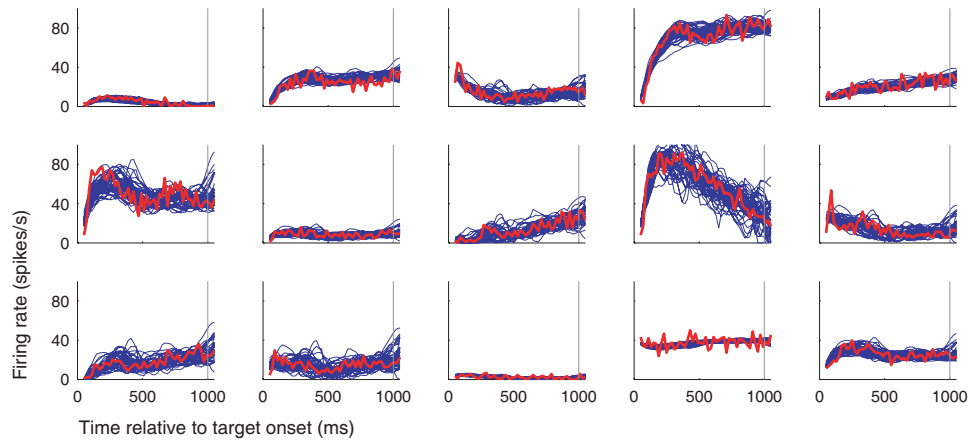

Figure 3: Imputed trial-by-trial firing rates (blue) and empirical firing rates (red). Gray vertical line indicates the time of the go cue. Each panel corresponds to one unit. For clarity, only test trials with delay periods of 1000 ms (44 trials) are plotted for each unit.

## Acknowledgments

This work was supported by NIH-NINDS-CRCNS-R01, NSF, NDSEGF, Gatsby, MSTP, CRPF, BWF, ONR, Sloan, and Whitaker. We would like to thank Dr. Mark Churchland for valuable discussions and Missy Howard for expert surgical assistance and veterinary care.

## References

[1] M. Abeles, H. Bergman, I. Gat, I. Meilijson, E. Seidemann, N. Tishby, and E. Vaadia. *Proc Natl Acad Sci USA*, 92:8616–8620, 1995.

[2] I. Gat, N. Tishby, and M. Abeles. *Network*, 8(3):297–322, 1997.

[3] A. Smith and E. Brown. *Neural Comput*, 15(5):965–991, 2003.

[4] S. Amari. *Biol Cybern*, 27(2):77–87, 1977.

[5] A. Dempster, N. Laird, and D. Rubin. *J R Stat Soc Ser B*, 39:1–38, 1977.

[6] S. Julier and J. Uhlmann. In *Proc. AeroSense: 11th Int. Symp. Aerospace/Defense Sensing, Simulation and Controls*, pp. 182–193, 1997.

[7] U. Lerner. *Hybrid Bayesian networks for reasoning about complex systems*. PhD thesis, Stanford University, Stanford, CA, 2002.

[8] J. Tanji and E. Evarts. *J Neurophysiol*, 39:1062–1068, 1976.

[9] M. Weinrich, S. Wise, and K. Mauritz. *Brain*, 107:385–414, 1984.

[10] M. Godschalk, R. Lemon, H. Kuypers, and J. van der Steen. *Behav Brain Res*, 18:143–157, 1985.

[11] A. Riehle and J. Requin. *J Neurophysiol*, 61:534–549, 1989.

[12] D. Crammond and J. Kalaska. *J Neurophysiol*, 84:986–1005, 2000.

[13] J. Messier and J. Kalaska. *J Neurophysiol*, 84:152–165, 2000.

[14] D. Rosenbaum. *J Exp Psychol Gen*, 109:444–474, 1980.

[15] A. Riehle and J. Requin. *J Behav Brain Res*, 53:35–49, 1993.

[16] M. Churchland, B. Yu, S. Ryu, G. Santhanam, and K. Shenoy. *Soc. for Neurosci. Abstr.*, 2004.

[17] E. Brown, R. Barbieri, V. Ventura, R. Kass, and L. Frank. *Neural Comput*, 14(2):325–346, 2002.
